# Characteristic Kernels on Groups and Semigroups

**Kenji Fukumizu**
Institute of Statistical Mathematics
4-6-7 Minami-Azabu, Minato-ku, Tokyo 106-8569 Japan
fukumizu@ism.ac.jp

**Bharath Sriperumbudur**
Department of ECE, UC San Diego
/ MPI for Biological Cybernetics
bharathsv@ucsd.edu

**Arthur Gretton**
MPI for Biological Cybernetics
Spemannstraße 38, 72076 Tübingen, Germany
arthur.gretton@tuebingen.mpg.de

**Bernhard Schölkopf**
MPI for Biological Cybernetics
bs@tuebingen.mpg.de

## Abstract

Embeddings of random variables in reproducing kernel Hilbert spaces (RKHSs) may be used to conduct statistical inference based on higher order moments. For sufficiently rich (characteristic) RKHSs, each probability distribution has a unique embedding, allowing all statistical properties of the distribution to be taken into consideration. Necessary and sufficient conditions for an RKHS to be characteristic exist for $\mathbb{R}^n$. In the present work, conditions are established for an RKHS to be characteristic on groups and semigroups. Illustrative examples are provided, including characteristic kernels on periodic domains, rotation matrices, and $\mathbb{R}_+^n$.

## 1 Introduction

Recent studies have shown that mapping random variables into a suitable reproducing kernel Hilbert space (RKHS) gives a powerful and straightforward method of dealing with higher-order statistics of the variables. For sufficiently rich RKHSs, it becomes possible to test whether two samples are from the same distribution, using the difference in their RKHS mappings [8]; as well as testing independence and conditional independence [6, 9]. It is also useful to optimize over kernel mappings on distributions, for instance to find the most predictive subspace in regression [5], or for ICA [1].

Key to the above work is the notion of a *characteristic kernel*, as introduced in [5, 6]: it gives an RKHS for which probabilities have unique images (*i.e.*, the mapping is injective). Such RKHSs are sufficiently rich in the sense required above. Universal kernels on compact metric spaces [16] are characteristic [8], as are Gaussian and Laplace kernels on $\mathbb{R}^n$ [6]. Recently, it has been shown [14] that a continuous shift-invariant $\mathbb{R}$-valued positive definite kernel on $\mathbb{R}^n$ is characteristic if and only if the support of its Fourier transform is the entire $\mathbb{R}^n$. This completely determines the set of characteristic ones in the convex cone of continuous shift-invariant positive definite kernels on $\mathbb{R}^n$.

One of the chief advantages of kernel methods is that they allow us to deal straightforwardly with complex domains, through use of a kernel function to determine the similarity between objects in these domains [13]. A question that naturally arises is whether characteristic kernels can be defined on spaces besides $\mathbb{R}^n$. Several such domains constitute topological groups/semigroups, and our focus is on kernels defined by their algebraic structure. Broadly speaking, our approach is based on extensions of Fourier analysis to groups and semigroups, where we apply appropriate extensions of Bochner's theorem to obtain the required conditions on the kernel.

The most immediate generalization of the results in [14] is to locally compact Abelian groups, of which $(\mathbb{R}^n, +)$ is one example. Thus, in Section 2 we provide review of characteristic kernels on $(\mathbb{R}^n, +)$ from this viewpoint. In Section 3 we derive necessary and sufficient conditions for kernels

on locally compact Abelian groups to be characteristic. Besides $(\mathbb{R}^n, +)$, such groups include $[0,1]^n$ with periodic boundary conditions [13, Section 4.4.4]. We next address non-Abelian compact groups in Section 4, for which we obtain a sufficient condition for a characteristic kernel. We illustrate with the example of $SO(3)$, which describes rotations in $\mathbb{R}^3$, and is used in fields such as geophysics [10] and robotics [15]. Finally, in Section 5, we consider the Abelian semigroup $(\mathbb{R}^n_+, +)$, where $\mathbb{R}_+ = [0, \infty)$. This semigroup has many practical applications, including expressions of nonnegative measures or frequency on $n$ points [3]. Note that in all cases, we provide specific examples of characteristic kernels to illustrate the properties required.

## 2  Preliminaries: Characteristic kernels and shift-invariant kernels

Let $X$ be a random variable taking values on a measurable space $(\Omega, \mathcal{B})$, and $\mathcal{H}$ be a RKHS defined by a measurable kernel $k$ on $\Omega$ such that $E[\sqrt{k(X,X)}] < \infty$. The *mean element* $m_X$ of $X$ is defined by the element in $\mathcal{H}$ such that $\langle m_X, f \rangle_{\mathcal{H}} = E[f(X)]$ $(\forall f \in \mathcal{H})$ (See [6, 7]). By plugging $f = k(\cdot, y)$ in the definition, the explicit functional form of $m_X$ is given by $m_X(y) = E[k(y, X)]$. A bounded measurable kernel $k$ on $\Omega$ is called *characteristic* if

$$\{P : \text{probability on } (\Omega, \mathcal{B})\} \to \mathcal{H}, \qquad P \mapsto m_P = E_{X \sim P}[k(\cdot, X)] \tag{1}$$

is injective ([5, 6]). Therefore, by definition, a characteristic kernel uniquely determines a probability by its mean element. This property is important in making inference on properties of distributions. It guarantees, for example, that $MMD = \|m_X - m_Y\|_{\mathcal{H}}$ is a (strict) distance on the space of probabilities on $\Omega$ [8]. The following result provides the necessary and sufficient condition for a kernel to be characteristic and shows its associated RKHS to be a rich function class.

**Lemma 1** ([7] Prop. 5). *Let $(\Omega, \mathcal{B})$ be a measurable space, $k$ be a bounded measurable positive definite kernel on $\Omega$, and $\mathcal{H}$ be the associated RKHS. Then, $k$ is characteristic if and only if $\mathcal{H} + \mathbb{R}$ (direct sum of the two RKHS's) is dense in $L^2(P)$ for every probability $P$ on $(\Omega, \mathcal{B})$.*

The above lemma and Theorem 3 of [6] imply that characteristic kernels give a criterion of (conditional) independence through (conditional) covariance on RKHS, which enables statistical tests of independence with kernels [6]. This explains also the practical importance of characteristic kernels.

The following result shows that the characteristic property is invariant under some conformal mappings introduced in [17] and provides a construction to generate new characteristic kernels.

**Lemma 2.** *Let $\Omega$ be a topological space with Borel $\sigma$-field, $k$ be a measurable positive definite kernel on $\Omega$ such that $\int_\Omega k(\cdot, y)d\mu(y) = 0$ means $\mu = 0$ for a finite Borel measure $\mu$, and $f : \Omega \to \mathbb{C}$ be a bounded continuous function such that $f(x) > 0$ for all $x \in \Omega$ and $k(x, x)|f(x)|^2$ is bounded. Then, the kernel $\tilde{k}(x, y) = f(x)k(x, y)f(y)$ is characteristic.*

*Proof.* Let $P$ and $Q$ be Borel probabilities such that $\int \tilde{k}(\cdot, x)dP(x) = \int \tilde{k}(\cdot, x)dQ(x)$. We have $\int k(\cdot, x)f(x)d(P - Q)(x) = 0$, which means $fP = fQ$. We have $P = Q$ by the positivity and continuity of $f$. $\qquad\square$

We will focus on spaces with algebraic structure for better description of characteristic kernels. Let $G$ be a group. A function $\phi : G \to \mathbb{C}$ is called *positive definite* if $k(x, y) = \phi(y^{-1}x)$ is a positive definite kernel. We call this type of positive definite kernels *shift-invariant*, because $k(zx, zy) = \phi((zy)^{-1}zx) = \phi(y^{-1}x) = k(x, y)$ for any $z \in G$.

There are many examples of shift-invariant positive definite kernels on the additive group $\mathbb{R}^n$: Gaussian RBF kernel $k(x, y) = \exp(-\|x-y\|^2/\sigma^2)$ and Laplacian kernel $k(x, y) = \exp(-\beta \sum_{i=1}^n |x_i - y_i|)$ are famous ones. In the case of $\mathbb{R}^n$, the following Bochner's theorem is well-known;

**Theorem 3** (Bochner). *Let $\phi : \mathbb{R}^n \to \mathbb{C}$ be a continuous function. $\phi$ is positive definite if and only if there is a unique finite non-negative Borel measure $\Lambda$ on $\mathbb{R}^n$ such that*

$$\phi(x) = \int_{\mathbb{R}^n} e^{\sqrt{-1}x^T \omega} d\Lambda(\omega). \tag{2}$$

Bochner's theorem completely characterizes the set of continuous shift-invariant positive definite kernels on $\mathbb{R}^n$ by the Fourier transform. It also implies that the continuous positive definite functions form a convex cone with the extreme points given by the Fourier kernels $\{e^{\sqrt{-1}x^T \omega} \mid \omega \in \mathbb{R}^n\}$.

It is interesting to determine the class of continuous shift-invariant "characteristic" kernels on $\mathbb{R}^n$. [14] gives a complete solution: if $\operatorname{supp}(\Lambda) = \mathbb{R}^n$,[1] then $\phi(x - y)$ is characteristic. In addition, if a continuous positive definite function of the form in Eq. (2) is real-valued and characteristic, then $\operatorname{supp}(\Lambda) = \mathbb{R}^n$. The basic idea is the following: since the mean element $E_P[\phi(y - X)]$ is equal to the convolution $\phi * P$, the Fourier transform rewrites the definition of characteristic property as

$$(\widehat{P} - \widehat{Q})\Lambda = 0 \quad \Longrightarrow \quad P = Q,$$

where $\widehat{\ }$ denotes the Fourier transform, and we use $\widehat{\phi * P} = \Lambda\widehat{P}$. Hence, it is natural to expect that if $\Lambda$ is everywhere positive, then $(\widehat{P} - \widehat{Q})$ must be zero, which means $P = Q$.

We will extend these results to more general algebraic objects, such as groups and semigroups, on which Fourier analysis and Bochner's theorem can be extended.

# 3   Characteristic kernels on locally compact Abelian groups

It is known that most of the results on Fourier analysis for $\mathbb{R}^n$ are extended to any locally compact Abelian (LCA) group, which is an Abelian (*i.e.* commutative) topological group with the topology Hausdorff and locally compact. The basic terminologies are provided in the supplementary material for readers who are not familiar to them. The group operation is denoted by "+" in Abelian cases.

Hereafter, for a LCA group $G$, we consider only the probability measures included in the set of finite regular measures $M(G)$ (see Supplements) to discuss characteristic property. This slightly restricts the class of measures, but removes only pathological ones.

## 3.1   Fourier analysis on LCA Group

We briefly summarize necessary results to show our main theorems. For the details, see [12, 11].

For a LCA group $G$, there exists a non-negative regular measure $m$ on $G$ such that $m(E + x) = m(E)$ for every $x \in G$ and every Borel set $E$ in $G$. This measure is called *Haar measure*. We use $dx$ to denote the Haar measure of $G$. With the Haar measure, the integral is shift-invariant, that is,

$$\int_G f(x + y)dx = \int_G f(x)dx \qquad (\forall y \in G).$$

The space of $L^p(G, dx)$ is simply denoted by $L^p(G)$.

A function $\gamma : G \to \mathbb{C}$ is called a *character* of $G$ if $\gamma(x + y) = \gamma(x)\gamma(y)$ and $|\gamma(x)| = 1$. The set of all continuous characters of $G$ forms an Abelian group with the operation $(\gamma_1\gamma_2)(x) = \gamma_1(x)\gamma_2(x)$. By convention, the group operation is denoted by addition "+", instead of multiplication; *i.e.*, $(\gamma_1 + \gamma_2)(x) = \gamma_1(x)\gamma_2(x)$. This group is called the *dual group* of $G$, and denoted by $\widehat{G}$.

For any $x \in G$, the function $\hat{x}$ on $\widehat{G}$ given by $\hat{x}(\gamma) = \gamma(x)$ $(\gamma \in \widehat{G})$ defines a character of $\widehat{G}$. It is known that $\widehat{G}$ is a LCA group if the weakest topology is introduced so that $\hat{x}$ is continuous for each $x \in G$. We can therefore consider the dual of $\widehat{G}$, denoted by $G^{\hat{}}$, and the group homomorphism

$$G \to G^{\hat{}}, \qquad x \mapsto \hat{x}.$$

The Pontryagin duality guarantees that this homomorphism is an isomorphism, and homeomorphism, thus $G^{\hat{}}$ can be identified with $G$. In view of the duality, it is customary to write $(x, \gamma) := \gamma(x)$. We have $(-x, \gamma) = (x, -\gamma) = \gamma(x)^{-1} = \overline{(x, \gamma)}$, where $\bar{z}$ is the complex conjugate of $z$.

Let $f \in L^1(G)$ and $\mu \in M(G)$, the Fourier transform of $f$ and $\mu$ are respectively defined by

$$\hat{f}(\gamma) = \int_G (-x, \gamma)f(x)dx, \qquad \hat{\mu}(\gamma) = \int_G (-x, \gamma)d\mu(x), \qquad (\gamma \in \widehat{G}). \qquad (3)$$

Let $f \in L^\infty(G)$, $g \in L^1(G)$, and $\mu, \nu \in M(G)$. The convolutions are defined respectively by

$$(g * f)(x) = \int_G f(x - y)g(y)dy, \quad (\mu * f)(x) = \int_G f(x - y)d\mu(y), \quad (\mu * \nu)(E) = \int_G \chi_E(x + y)d\mu(x)d\nu(y).$$

$g * f$ is uniformly continuous on $G$. For any $f, g \in L^1(G)$ and $\mu, \nu \in M(G)$, we have the formula

$$\widehat{f * g} = \hat{f}\hat{g}, \qquad \widehat{\mu * f} = \hat{\mu}\hat{f}, \qquad \widehat{\mu * \nu} = \hat{\mu}\hat{\nu}. \tag{4}$$

The following facts are basic ( [12], Section 1.3).

**Proposition 4.** *For $\mu \in M(G)$, the Fourier transform $\hat{\mu}$ is bounded and uniformly continuous.*

**Theorem 5** (Uniqueness theorem). *If $\mu \in M(G)$ satisfies $\widehat{\mu} = 0$, then $\mu = 0$.*

It is known that the dual group of the LCA group $\mathbb{R}^n$ is $\{e^{\sqrt{-1}\omega^T x} \mid \omega \in \mathbb{R}^n\}$, which can be identified with $\mathbb{R}^n$. The above definition and properties of Fourier transform for LCA groups are extension of the ordinary Fourier transform for $\mathbb{R}^n$. Bochner's theorem can be also extended.

**Theorem 6** (Bochner's theorem. *e.g.*, [12] Section 1.4.3). *A continuous function $\phi$ on $G$ is positive definite if and only if there is a unique non-negative measure $\Lambda \in M(\widehat{G})$ such that*

$$\phi(x) = \int_{\widehat{G}} (x, \gamma) d\Lambda(\gamma) \qquad (x \in G). \tag{5}$$

## 3.2 Shift-invariant characteristic kernels on LCA group

Based on Bochner's theorem, a sufficient condition of the characteristic property is obtained.

**Theorem 7.** *Let $\phi$ be a continuous positive definite function on a LCA group $G$ given by Eq. (5) with $\Lambda$. If $supp(\Lambda) = \widehat{G}$, then the positive definite kernel $k(x, y) = \phi(x - y)$ is characteristic.*

*Proof.* It suffices to prove that if $\mu \in M(G)$ satisfies $\mu * \phi = 0$ then $\mu = 0$. We have $\int_G (\mu * \phi)(x) d\mu(x) = 0$. On the other hand, by using Fubini's theorem,

$$\int_G (\mu * \phi)(x) d\mu(x) = \int_G \int_G \phi(x - y) d\mu(y) d\mu(x) = \int_G \int_G \int_{\widehat{G}} (x - y, \gamma) d\Lambda(\gamma) d\mu(y) d\mu(x)$$
$$= \int_{\widehat{G}} \int_G (x, \gamma) d\mu(x) \int_G (-y, \gamma) d\mu(y) d\Lambda(\gamma) = \int_{\widehat{G}} |\widehat{\mu}(\gamma)|^2 d\Lambda(\gamma).$$

Since $\widehat{\mu}$ is continuous and $supp(\Lambda) = \widehat{G}$, we have $\widehat{\mu} = 0$, which means $\mu = 0$ by Theorem 5. $\square$

In real-valued cases, the condition $supp(\Lambda) = \widehat{G}$ is almost necessary.

**Theorem 8.** *Let $\phi$ be a $\mathbb{R}$-valued continuous positive definite function on a LCA group $G$ given by Eq. (5) with $\Lambda$. The kernel $\phi(x - y)$ is characteristic if and only if (i) $0 \in \widehat{G}$ is not open and $supp(\Lambda) = \widehat{G}$, or (ii) $0 \in \widehat{G}$ is open and $supp(\Lambda) \supset \widehat{G} - \{0\}$. The case (ii) occurs if $G$ is compact.*

*Proof.* It suffices to prove the only if part. Assume $k(x, y) = \phi(x - y)$ is characteristic. It is obvious that $k$ is characteristic if and only if so is $k(x, y) + 1$. Thus, we can assume $0 \in supp(\Lambda)$. Suppose $supp(\Lambda) \neq \widehat{G}$. Since $\phi$ is real-valued, $\Lambda(-E) = \Lambda(E)$ for every Borel set $E$. Thus $U := \widehat{G} \backslash supp(\Lambda)$ is a non-empty open set, with $-U = U$, and $0 \notin U$ by assumption. Let $\gamma_0 \in U$ and $\tau : \widehat{G} \times \widehat{G} \to \widehat{G}, (\gamma_1, \gamma_2) \mapsto \gamma_1 - \gamma_2$. Take an open neighborhood $W$ of $0$ in $\widehat{G}$ with compact closure such that $W \subset \tau^{-1}(U - \gamma_0)$. Then, $(W + (-W) + \gamma_0) \cup (W + (-W) - \gamma_0) \subset U$.

Let $g = \chi_W * \chi_{-W}$, where $\chi_E$ denotes the indicator function of a set $E$. $g$ is continuous, and $supp(g) \subset cl(W + (-W))$. Also, $g$ is positive definite, since $\sum_{i,j} c_i \overline{c_j} g(x_i - x_j) = \sum_{i,j} c_i \overline{c_j} \int_G \chi_W(x_i - x_j - y) \chi_{-W}(y) dy = \sum_{i,j} c_i \overline{c_j} \int_G \chi_W(x_i - y) \chi_{-W}(y - x_j) dy = \int_G \left( \sum_i c_i \chi_W(x_i - y) \right) \left( \overline{\sum_j c_j \chi_W(x_j - y)} \right) dy \geq 0$. By Bochner's theorem and Pontryagin duality, there is a non-negative measure $\mu \in M(G)$ such that

$$g(\gamma) = \int_G (x, \gamma) d\mu(x) \qquad (\gamma \in \widehat{G}).$$

It follows that $g(\gamma - \gamma_0) + g(\gamma + \gamma_0) = \int_G \{(x, \gamma - \gamma_0) + (x, \gamma + \gamma_0)\} d\mu(x) = \int_G (x, \gamma) d((\gamma_0 + \overline{\gamma_0})\mu)(x)$.

Since $supp(g) \subset cl(W + (-W))$, the left hand side is non-zero only in $(W + (-W) + \gamma_0) \cup (W + (-W) - \gamma_0) \subset U$, which does not contain $0$. Thus, by setting $\gamma = 0$, we have

$$((\gamma_0 + \overline{\gamma_0})\mu)(G) = 0. \tag{6}$$

The measure $(\gamma_0 + \overline{\gamma_0})\mu$ is real-valued, and non-zero since the function $g(\gamma - \gamma_0) + g(\gamma + \gamma_0)$ is not constant zero. Let $m = |(\gamma_0 + \overline{\gamma_0})\mu|(G)$, and define the non-negative measures

$$\mu_1 = |(\gamma_0 + \overline{\gamma_0})\mu|/m, \qquad \mu_2 = \{|(\gamma_0 + \overline{\gamma_0})\mu| - (\gamma_0 + \overline{\gamma_0})\mu\}/m.$$

Both of $\mu_1$ and $\mu_2$ are probability measures on $G$ from Eq. (6), and $\mu_1 \neq \mu_2$. From Fubini's theorem,

$m \times ((\mu_1 - \mu_2) * \phi)(x) = \int_G \phi(x - y)(\gamma_0(y) + \overline{\gamma_0}(y))d\mu(y)$

$= \int_{\widehat{G}} (x, \gamma) \int_G \overline{\{(y, \gamma - \gamma_0) + (y, \gamma + \gamma_0)\}} d\mu(y) d\Lambda(\gamma) = \int_{\widehat{G}} (x, \gamma)\{g(\gamma - \gamma_0) + g(\gamma + \gamma_0)\} d\Lambda(\gamma)$

Since the integrand is zero in $\operatorname{supp}(\Lambda)$, we have $(\mu_1 - \mu_2) * \phi = 0$, which derives contradiction. The last assertion is obvious, since $\widehat{G}$ is discrete if and only if $G$ is compact [12, Sec. 1.7.3]. $\qquad\square$

Theorems 7 and 8 are generalization of the results in [14]. From Theorem 8, we can see that the characteristic property is stable under the product for shift-invariant kernels.

**Corollary 9.** *Let $\phi_1(x - y)$ and $\phi_2(x - y)$ be $\mathbb{R}$-valued continuous shift-invariant characteristic kernels on a LCA group $G$. If (i) $G$ is non-compact, or (ii) $G$ is compact and $2\gamma \neq 0$ for any nonzero $\gamma \in \widehat{G}$. Then $(\phi_1\phi_2)(x - y)$ is characteristic.*

*Proof.* We show the proof only for (i). Let $\Lambda_1, \Lambda_2$ be the non-negative measures to give $\phi_1$ and $\phi_2$, respectively, in Eq. (5). By Theorem 8, $\operatorname{supp}(\Lambda_1) = \operatorname{supp}(\Lambda_2) = \widehat{G}$. This means $\operatorname{supp}(\Lambda_1 * \Lambda_2) = \widehat{G}$. The proof is completed because $\Lambda_1 * \Lambda_2$ gives a positive definite function $\phi_1\phi_2$. $\qquad\square$

**Example 1.** $(\mathbb{R}^n, +)$: As already shown in [6, 14], the Gaussian RBF kernel $\exp(-\frac{1}{2\sigma^2}\|x - y\|^2)$ and Laplacian kernel $\exp(-\beta \sum_{i=1}^n |x_i - y_i|)$ are characteristic on $\mathbb{R}^n$. An example of a positive definite kernel that is *not* characteristic on $\mathbb{R}^n$ is $\operatorname{sinc}(x - y) = \frac{\sin(x-y)}{x-y}$.

**Example 2.** $([0, 2\pi), +)$: The addition is made modulo $2\pi$. The dual group is $\{e^{\sqrt{-1}nx} \mid n \in \mathbb{Z}\}$, which is isomorphic to $\mathbb{Z}$. The Fourier transform is equal to the ordinary Fourier expansion. The following are examples of characteristic kernels given by the expression

$$\phi(x) = \sum_{n=-\infty}^{\infty} a_n e^{\sqrt{-1}nx}, \qquad a_0 \geq 0, \ a_n > 0 \ (n \neq 0), \ \sum_{n=0}^{\infty} a_n < \infty.$$

(1) $a_0 = \pi^2/3, a_n = 2/n^2 \ (n \neq 0)$ $\qquad \Rightarrow \qquad k_1(x, y) = (\pi - (x - y)_{mod\ 2\pi})^2.$

(2) $a_0 = 1/2, a_n = 1/(1 + n^2) \ (n \neq 0)$ $\qquad \Rightarrow \qquad k_2(x, y) = \cosh(\pi - (x - y)_{mod\ 2\pi}).$

(3) $a_0 = 0, \ a_n = \alpha^n/n \ (n \neq 0), \ (|\alpha| < 1) \quad \Rightarrow \quad k_3(x, y) = -\log(1 - 2\alpha\cos(x - y) + \alpha^2).$

(4) $a_n = \alpha^{|n|}, \ (0 < \alpha < 1) \quad \Rightarrow \quad k_4(x, y) = 1/(1 - 2\alpha\cos(x - y) + \alpha^2)$ (Poisson kernel).

Examples of *non*-characteristic kernels on $[0, 2\pi)$ include $\cos(x - y)$, Féjer, and Dirichlet kernel.

# 4  Characteristic kernels on compact groups

We discuss non-Abelian cases in this section. Non-Abelian groups include various matrix groups, such as $SO(3) = \{A \in M(3 \times 3; \mathbb{R}) \mid A^T A = I_3, \det A = 1\}$, which represents rotations in $\mathbb{R}^3$. $SO(3)$ is used in practice as the data space of rotational data, which popularly appear in many fields such as geophysics [10] and robotics [15]. Providing useful positive definite kernels on this class is important in those applications areas. First, we give a brief summary of known results on the Fourier analysis on locally compact and compact groups. See [11, 4] for the details.

## 4.1  Unitary representation and Fourier analysis

Let $G$ be a locally compact group, which may not be Abelian. A *unitary representation* $(T, H)$ of $G$ is a group homomorphism $T$ into the group $U(H)$ of unitary operators on some nonzero Hilbert space $H$, that is, a map $T : G \to U(H)$ that satisfies $T(xy) = T(x)T(y)$ and $T(x^{-1}) = T(x)^{-1} = T(x)^*$, and for which $x \mapsto T(x)u$ is continuous from $G$ to $H$ for any $u \in H$.

For a unitary representation $(T, H)$ on a locally compact group $G$, a subspace $V$ in $H$ is called $G$-invariant if $T(x)V \subset V$ for every $x \in G$. A unitary representation $(T, H)$ is *irreducible* if there are

no closed $G$-invariant subspace except $\{0\}$ and $H$. Unitary representations $(T_1, H_1)$ and $(T_2, H_2)$ are said to be *equivalent* if there is a unitary isomorphism $A : H_1 \to H_2$ such that $T_1 = A^{-1}T_2A$.

The following facts are basic (e.g., [4], Section 3,1, 5.1).

**Theorem 10.** *(i) If $G$ is a compact group, every irreducible unitary representation $(T, H)$ of $G$ is finite dimensional, that is, $H$ is finite dimensional. (ii) If $G$ is an Abelian group, every irreducible unitary representation of $G$ is one dimensional. They are the continuous characters of $G$.*

It is possible to extend the Fourier analysis on locally compact non-Abelian groups. Unlike Abelian cases, the Fourier transform by the characters are not possible, but we need to consider unitary representations and operator-valued Fourier transform. Since extending the results of the LCA case to the general cases causes very complicated topology, we focus on compact groups. Also, for simplicity, we assume that $G$ is second countable, *i.e.*, there are countable open basis on $G$.

We define $\widehat{G}$ to be the set of equivalent classes of irreducible unitary representations of a compact group $G$. The equivalence class of a unitary representation $(T, H_T)$ is denoted by $[T]$, and the dimensionality of $H_T$ by $d_T$. We fix a representative $T$ for every $[T] \in \widehat{G}$ for all.

It is known that on a compact group $G$ there is a Haar measure $m$, which is a left and right invariant non-negative finite measure. We normalize it so that $m(G) = 1$ and denote it by $dx$.

Let $(T, H_T)$ be a unitary representation. For $f \in L^1(G)$ and $\mu \in M(G)$, the Fourier transform of $f$ and $\mu$ are defined by the "operator-valued" functions on $\widehat{G}$,

$$\widehat{f}(T) = \int_G f(x)T(x^{-1})dx = \int_G f(x)T(x)^*dx, \quad \widehat{\mu}(T) = \int_G T(x^{-1})d\mu(x) = \int_G T(x)^*d\mu(x),$$

respectively. These are operators on $H_T$. This is a natural extension of the Fourier transform on LCA groups, where $\widehat{G}$ is the characters serving as the Fourier kernel in view of Theorem 10.

We can define the "inverse Fourier transform". Let $A_T$ ($[T] \in \widehat{G}$) be an operator on $H_T$. The series

$$\sum_{[T]\in\widehat{G}} d_T \mathrm{Tr}[A_T T(x)] \tag{7}$$

is said to be *absolutely convergent* if $\sum_{[T]\in\widehat{G}} d_T \mathrm{Tr}[|A_T|] < \infty$, where $|A| = \sqrt{A^TA}$. It is obvious that if the above series is absolutely convergent, the convergence is uniform on $G$. It is known that if $G$ is second countable, $\widehat{G}$ is at most countable, thus the sum is taken over the countable set.

Bochner's theorem can be extended to compact groups as follows [11, Section 34.10].

**Theorem 11.** *A continuous function $\phi$ on a compact group $G$ is positive definite if and only if the Fourier transform $\widehat{\phi}(T)$ is positive semidefinite, gives an absolutely convergent series Eq. (7), and*

$$\phi(x) = \sum_{[T]\in\widehat{G}} d_T \mathrm{Tr}[\widehat{\phi}(T)T(x)]. \tag{8}$$

The proof of "if" part is easy; in fact, $\sum_{i,j} c_i\overline{c_j}\phi(x_j^{-1}x_i) = \sum_{i,j} c_i\overline{c_j}\sum_{[T]\in\widehat{G}} d_T \mathrm{Tr}[\widehat{\phi}(T)T(x_j^{-1}x_i)]$
$= \sum_{i,j} c_i\overline{c_j}\sum_{[T]} d_T \mathrm{Tr}[T(x_i)\widehat{\phi}(T)T(x_j)^*] = \sum_{[T]} d_T \mathrm{Tr}[\left(\sum_i c_i T(x_i)\right)\widehat{\phi}(T)\left(\sum_j c_j T(x_j)\right)^*] \geq 0$.

## 4.2 Shift-invariant characteristic kernels on compact groups

We have the following sufficient condition of characteristic property for compact groups.

**Theorem 12.** *Let $\phi$ be a positive definite function of the form Eq. (8) on a compact group $G$. If $\widehat{\phi}(T)$ is strictly positive definite for every $[T] \in \widehat{G}\backslash\{1\}$, the kernel $\phi(y^{-1}x)$ is characteristic.*

*Proof.* Let $P, Q \in M(G)$ be probabilities on $G$. Define $\mu = P - Q$, and suppose $\int_G \phi(y^{-1}x)d\mu(y) = 0$. If we take the integral over $x$ with the measure $\mu$, Fubini's theorem shows $0 = \int_G \int_G \sum_{[T]} d_T \mathrm{Tr}[\widehat{\phi}(T)T(y^{-1}x)]d\mu(y)d\mu(x) = \sum_{[T]} d_T \int_G \int_G \mathrm{Tr}[T(x)\widehat{\phi}(T)T(y)^*]d\mu(x)d\mu(y) = \sum_{[T]} d_T \mathrm{Tr}[\widehat{\mu}(T)\widehat{\phi}(T)\widehat{\mu}(T)^*]$. Since $d_T > 0$ and $\widehat{\phi}(T)$ is strictly positive, $\widehat{\mu}(T) = 0$ for every $[T] \in \widehat{G}$, that is, $\int_G T(x)^*d\mu(x) = O$. If we fix an orthonormal basis of $H_T$ and express $T(x)$ by the matrix elements $T_{ij}(x)$, we have

$$\int_G T_{ij}(x)d\mu(x) = 0 \qquad (\forall [T] \in \widehat{G}, i, j = 1, \ldots, d_T).$$

The Peter-Weyl Theorem (e.g., [4, Section 5.2]) shows that $\{\sqrt{d_T}T_{ij}(x) \mid [T] \in \widehat{G}, i, j = 1, \ldots, d_T\}$ is a complete orthonormal basis of $L^2(G)$, which means $\mu = 0$. $\qquad\square$

It is interesting to ask whether Theorem 8 can be extended to compact groups. The same proof does not apply, however, because application of Bochner's theorem to a positive definite function on $\widehat{G}$ is not possible by the lack of duality.

**Example of SO(3).**     It is known that $\widehat{SO(3)}$ consists of $(T_n, H_n)$ $(n = 0, 1, 2, \ldots)$, where $d_{T_n} = 2n + 1$. We omit the explicit form of $T_n$, while it is known (e.g., [4], Section 5.4), but use the *character* defined by $\gamma_n(x) = \text{Tr}[T_n(x)]$. It is also known that $\gamma_n$ is given by

$$\gamma_n(A) = \frac{\sin((2n+1)\theta)}{\sin\theta} \qquad (n = 0, 1, 2, \ldots),$$

where $e^{\pm\sqrt{-1}\theta}$ $(0 \leq \theta \leq \pi)$ are the eigenvalues of $A$, *i.e.*, $\cos\theta = \frac{1}{2}\text{Tr}[A]$. Since plugging $\widehat{\phi}(T_n) = a_n I_{d_{T_n}}$ in Eq. (8) derives $a_n\gamma_n$ for each term, we see that a sequence $\{a_n\}_{n=0}^{\infty}$ such that $a_0 \geq 0$, $a_n > 0$ $(n \geq 1)$, and $\sum_{n=0}^{\infty} a_n(2n+1)^2 < \infty$ defines a characteristic positive definite kernel on $SO(3)$ by

$$k(A, B) = \sum_{n=0}^{\infty}(2n+1)a_n\frac{\sin((2n+1)\theta)}{\sin\theta} \qquad (\cos\theta = \frac{1}{2}\text{Tr}[B^{-1}A], \ 0 \leq \theta \leq \pi).$$

Some examples are listed below ($\alpha$ is a parameter such that $|\alpha| < 1$).

(1)  $a_n = \dfrac{1}{(2n+1)^4}$ :       $k_1(A, B) = \dfrac{1}{\sin\theta}\sum_{n=0}^{\infty}\dfrac{\sin((2n+1)\theta)}{(2n+1)^3} = \dfrac{\pi\theta(\pi-\theta)}{8\sin\theta}.$

(2)  $a_n = \dfrac{\alpha^{2n+1}}{(2n+1)^2}$ :      $k_2(A, B) = \displaystyle\sum_{n=0}^{\infty}\dfrac{\alpha^{2n+1}\sin((2n+1)\theta)}{(2n+1)\sin\theta} = \dfrac{1}{2\sin\theta}\arctan\left(\dfrac{2\alpha\sin\theta}{1-\alpha^2}\right).$

# 5   Characteristic kernels on the semigroup $\mathbb{R}_+^n$

In this section, we consider kernels on an Abelian semigroup $(S, +)$. In this case, a kernel based on the semigroup structure is defined by $k(x, y) = \phi(x + y)$. For an Abelian semigroup $(S, +)$, a semicharacter is defined by a map $\rho : S \to \mathbb{C}$ such that $\rho(x + y) = \rho(x)\rho(y)$.

While extensions of Bochner's theorem are known for semigroups [2], the topology on the set of semicharacters are not as obvious as LCA groups, and the straightforward extension of the results in Section 3 is difficult. We focus only on the Abelian semigroup $(\mathbb{R}_+^n, +)$, where $\mathbb{R}_+ = [0, \infty)$. This semigroup has many practical applications of data analysis including expressions of nonnegative measures or frequency on $n$ points [3]. For $\mathbb{R}_+^n$, it is easy to see the bounded continuous semicharacters are given by $\{\prod_{i=1}^{n} e^{-\lambda_i x} \mid \lambda_i \geq 0 \ (i = 1, \ldots, n)\}$ [2, Section 4.4].

For $\mathbb{R}_+^n$, Laplace transform replaces Fourier transform to give Bochner's theorem.

**Theorem 13** ([2], Section 4.4). *Let $\phi$ be a bounded continuous function on $\mathbb{R}_+^n$. $\phi$ is positive definite if and only if there exists a unique non-negative measure $\Lambda \in M(\mathbb{R}_+^n)$ such that*

$$\phi(x) = \int_{\mathbb{R}_+^n} e^{-\sum_{i=1}^{n} t_i x_i} d\Lambda(t) \qquad (\forall x \in \mathbb{R}_+^n). \tag{9}$$

Based on the above theorem, we have the following sufficient condition of characteristic property.

**Theorem 14.** *Let $\phi$ be a positive definite function given by Eq. (9). If $\text{supp}\Lambda = \mathbb{R}_+^n$, then the positive definite kernel $k(x, y) = \phi(x + y)$ is characteristic.*

*Proof.* Let $P$ and $Q$ be probabilities on $\mathbb{R}_+^n$, and $\mu = P - Q$. Define the Laplace transform by $\mathcal{L}\mu(t) = \int_{\mathbb{R}_+^n} e^{-\sum_{i=1}^{n} t_i x_i} d\mu(x)$. It is easy to see $\mathcal{L}\mu$ is bounded and continuous on $\mathbb{R}_+^n$. Suppose $\int \phi(x + y)d\mu(y) = 0$ for all $x \in \mathbb{R}_+^n$. In exactly the same way as the proof of Theorem 7, we have $\mathcal{L}P = \mathcal{L}Q$. By the uniqueness part of Theorem 13, we conclude $P = Q$. $\qquad\square$

We show some examples of characteristic kernels on $(\mathbb{R}_+^n, +)$. Let $a = (a_i)_{i=1}^n$ and $b = (b_i)_{i=1}^n$ $(a_i \geq 0, b_i \geq 0)$ be non-negative measures on $n$ points.

(1) $\quad \Lambda = \prod_{i=1}^n t_i^{\nu-1} e^{\lambda t_i} \ (\lambda > 0):$ $\qquad\qquad k_1(a,b) = \prod_{i=1}^n (a_i + b_i + \lambda)^{-1}.$

(2) $\quad \Lambda = t^{-3/2} e^{-\beta^2/(4t)} \ (\beta > 0):$ $\qquad\qquad k_2(a,b) = e^{-\beta \sum_{i=1}^n \sqrt{a_i + b_i}}.$

Since the proof of Theorem 14 shows $\int \phi(x+y) d\mu(y) = 0$ means $\mu = 0$ for $\mu \in M(\mathbb{R}_+^n)$, Lemma 2 shows

$$\tilde{k}_2(a,b) = \exp\big\{ -\beta \big( \sum_{i=1}^n \sqrt{(a_i+b_i)/2} - (\sum_{i=1}^n \sqrt{a_i} + \sum_{i=1}^n \sqrt{b_i})/2 \big) \big\}$$

is also characteristic. The exponent has the form $h\big(\frac{a+b}{2}\big) - \frac{h(a)+h(b)}{2}$ with $h(c) = \sum_{i=1}^n \sqrt{c_i}$, which compares the value of $h$ of the merged measure $(a+b)/2$ and the average of $h(a)$ and $h(b)$. This type of kernel on non-negative measures is discussed in [3] in connection with semigroup structure.

## 6   Conclusions

We have discussed conditions that kernels defined by the algebraic structure of groups and semi-groups are characteristic. For locally compact Abelian groups, the continuous shift-invariant $\mathbb{R}$-valued characteristic kernels are completely determined by the Fourier inverse of positive measures with support equal to the entire dual group. For compact (non-Abelian) groups, we show a sufficient condition of continuous shift-invariant characteristic kernels in terms of the operator-valued Fourier transform. We show a condition for the semigroup $\mathbb{R}_+^n$. In the advanced theory of harmonic analysis, Bochner's theorem and Fourier analysis can be extended to more general algebraic structure to some extent. It is interesting to consider generalization of the results in this paper to such general classes.

In practical applications of machine learning, we are given a finite sample from a distribution, rather than the distribution itself. In this setting, it becomes important to choose the best possible kernel for inference on this sample. While the characteristic property gives a necessary requirement for RKHS embeddings of distributions to be distinguishable, it does not address optimal kernel choice at finite sample sizes. Theoretical approaches to this problem are the basis for future work.

## Footnotes

[1]For a finite regular measure, there is the largest open set $U$ with $\mu(U) = 0$. The complement of $U$ is called the *support* of $\mu$, and denoted by $\operatorname{supp}(\mu)$. See the supplementary material for the detail.

## References

[1] F. R. Bach and M. I. Jordan. Kernel independent component analysis. *JMLR*, 3:1–48, 2002.

[2] C. Berg, J. P. R. Christensen, and P. Ressel. *Harmonic Analysis on Semigroups*. Springer, 1984.

[3] M. Cuturi, K. Fukumizu, and J.-P. Vert. Semigroup kernels on measures. *JMLR*, 6:1169–1198, 2005.

[4] B. B. Folland. *A course in abstract harmonic analysis*. CRC Press, 1995.

[5] K. Fukumizu, F. R. Bach, and M. I. Jordan. Dimensionality reduction for supervised learning with reproducing kernel Hilbert spaces. *JMLR*, 5:73–99, 2004.

[6] K. Fukumizu, A. Gretton, X. Sun, and B. Schölkopf. Kernel measures of conditional dependence. *Advances in NIPS 20*, 489–496. MIT Press, 2008.

[7] K. Fukumizu, F. R.Bach, and M. I. Jordan. Kernel dimension reduction in regression. *The Annals of Statistics*, 2009, in press.

[8] A. Gretton, K. M. Borgwardt, M. Rasch, B. Schölkopf, and A. Smola. A kernel method for the two-sample-problem. *Advances in NIPS 19*. MIT Press, 2007.

[9] A. Gretton, K. Fukumizu, C. H. Teo, L. Song, B. Schölkopf, and A. Smola. A kernel statistical test of independence. *Advances in NIPS 20*, 585–592. MIT Press, 2008.

[10] M. S. Hanna and T. Chang. Fitting smooth histories to rotation data. *Journal of Multivariate Analysis*, 75:47–61, 2000.

[11] E. Hewitt and K. A. Ross. *Abstract Harmonic Analysis II*. 1970.

[12] W. Rudin. *Fourier Analysis on Groups*. Interscience, 1962.

[13] B. Schölkopf and A.J. Smola. *Learning with Kernels*. MIT Press. 2002.

[14] B. K. Sriperumbudur, A. Gretton, K. Fukumizu, G. Lanckriet, and B. Schölkopf. Injective Hilbert space embeddings of probability measures. In *Proc. COLT 2008*, to appear, 2008.

[15] O. Stavdahl, A. K. Bondhus, K. Y. Pettersen, and K. E. Malvig. Optimal statistical operators for 3-dimensional rotational data: geometric interpretations and application to prosthesis kinematics. *Robotica*, 23(3):283–292, 2005.

[16] I. Steinwart. On the influence of the kernel on the consistency of support vector machines. *JMLR*, 2:67–93, 2001.

[17] S. Wu and S-I. Amari. Conformal Transformation of Kernel Functions: A Data-Dependent Way to Improve Support Vector Machine Classifiers. *Neural Process. Lett.*, 15(1):59–67, 2002.

